# Kernel Measures of Conditional Dependence

**Kenji Fukumizu**
Institute of Statistical Mathematics
4-6-7 Minami-Azabu, Minato-ku
Tokyo 106-8569 Japan
fukumizu@ism.ac.jp

**Arthur Gretton**
Max-Planck Institute for Biological Cybernetics
Spemannstraße 38, 72076 Tübingen, Germany
arthur.gretton@tuebingen.mpg.de

**Xiaohai Sun**
Max-Planck Institute for Biological Cybernetics
Spemannstraße 38, 72076 Tübingen, Germany
xiaohi@tuebingen.mpg.de

**Bernhard Schölkopf**
Max-Planck Institute for Biological Cybernetics
Spemannstraße 38, 72076 Tübingen, Germany
bernhard.schoelkopf@tuebingen.mpg.de

## Abstract

We propose a new measure of conditional dependence of random variables, based on normalized cross-covariance operators on reproducing kernel Hilbert spaces. Unlike previous kernel dependence measures, the proposed criterion does not depend on the choice of kernel in the limit of infinite data, for a wide class of kernels. At the same time, it has a straightforward empirical estimate with good convergence behaviour. We discuss the theoretical properties of the measure, and demonstrate its application in experiments.

## 1 Introduction

Measuring dependence of random variables is one of the main concerns of statistical inference. A typical example is the inference of a graphical model, which expresses the relations among variables in terms of independence and conditional independence. Independent component analysis employs a measure of independence as the objective function, and feature selection in supervised learning looks for a set of features on which the response variable most depends.

Kernel methods have been successfully used for capturing (conditional) dependence of variables [1, 5, 8, 9, 16]. With the ability to represent high order moments, mapping of variables into reproducing kernel Hilbert spaces (RKHSs) allows us to infer properties of the distributions, such as independence and homogeneity [7]. A drawback of previous kernel dependence measures, however, is that their value depends not only on the distribution of the variables, but also on the kernel, in contrast to measures such as mutual information.

In this paper, we propose to use the Hilbert-Schmidt norm of the normalized conditional cross-covariance operator, and show that this operator encodes the dependence structure of random variables. Our criterion includes a measure of unconditional dependence as a special case. We prove in the limit of infinite data, under assumptions on the richness of the RKHS, that this measure has an explicit integral expression which depends only on the probability densities of the variables, despite being defined in terms of kernels. We also prove that its empirical estimate converges to the kernel-independent value as the sample size increases. Furthermore, we provide a general formulation for

the "richness" of an RKHS, and a theoretically motivated kernel selection method. We successfully apply our measure in experiments on synthetic and real data.

## 2    Measuring conditional dependence with kernels

The probability law of a random variable $X$ is denoted by $P_X$, and the space of the square integrable functions with probability $P$ by $L^2(P)$. The symbol $X \perp\!\!\!\perp Y \mid Z$ indicates the conditional independence of $X$ and $Y$ given $Z$. The null space and the range of an operator $T$ are written $\mathcal{N}(T)$ and $\mathcal{R}(T)$, respectively.

### 2.1    Dependence measures with normalized cross-covariance operators

Covariance operators on RKHSs have been successfully used for capturing dependence and conditional dependence of random variables, by incorporating high order moments [5, 8, 16]. We give a brief review here; see [5, 6, 2] for further detail. Suppose we have a random variable $(X, Y)$ on $\mathcal{X} \times \mathcal{Y}$, and RKHSs $\mathcal{H}_\mathcal{X}$ and $\mathcal{H}_\mathcal{Y}$ on $\mathcal{X}$ and $\mathcal{Y}$, respectively, with measurable positive definite kernels $k_\mathcal{X}$ and $k_\mathcal{Y}$. Throughout this paper, we assume the integrability

(A-1)                    $E[k_\mathcal{X}(X, X)] < \infty, \quad E[k_\mathcal{Y}(Y, Y)] < \infty.$

This assumption ensures $\mathcal{H}_\mathcal{X} \subset L^2(P_X)$ and $\mathcal{H}_\mathcal{Y} \subset L^2(P_Y)$. The *cross-covariance operator* $\Sigma_{YX} : \mathcal{H}_\mathcal{X} \to \mathcal{H}_\mathcal{Y}$ is defined by the unique bounded operator that satisfies

$$\langle g, \Sigma_{YX} f \rangle_{\mathcal{H}_\mathcal{Y}} = \mathrm{Cov}[f(X), g(Y)] \quad (= E[f(X)g(Y)] - E[f(X)]E[g(Y)]) \tag{1}$$

for all $f \in \mathcal{H}_\mathcal{X}$ and $g \in \mathcal{H}_\mathcal{Y}$. If $Y = X$, $\Sigma_{XX}$ is called the covariance operator, which is self-adjoint and positive. The operator $\Sigma_{YX}$ naturally extends the covariance matrix $C_{YX}$ on Euclidean spaces, and represents higher order correlations of $X$ and $Y$ through $f(X)$ and $g(Y)$ with nonlinear kernels.

It is known [2] that the cross-covariance operator can be decomposed into the covariance of the marginals and the correlation; that is, there exists a unique bounded operator $V_{YX}$ such that

$$\Sigma_{YX} = \Sigma_{YY}^{1/2} V_{YX} \Sigma_{XX}^{1/2}, \tag{2}$$

$\mathcal{R}(V_{YX}) \subset \overline{\mathcal{R}(\Sigma_{YY})}$, and $\mathcal{N}(V_{YX})^\perp \subset \overline{\mathcal{R}(\Sigma_{XX})}$. The operator norm of $V_{YX}$ is less than or equal to 1. We call $V_{YX}$ the *normalized cross-covariance operator* (NOCCO, see also [4]).

While the operator $V_{YX}$ encodes the same information regarding the dependence of $X$ and $Y$ as $\Sigma_{YX}$, the former rather expresses the information more directly than $\Sigma_{YX}$, with less influence of the marginals. This relation can be understood as an analogue to the difference between the covariance $\mathrm{Cov}[X, Y]$ and the correlation $\mathrm{Cov}[X, Y]/(\mathrm{Var}(X)\mathrm{Var}(Y))^{1/2}$. Note also that kernel canonical correlation analysis [1] uses the largest eigenvalue of $V_{YX}$ and its corresponding eigenfunctions [4].

Suppose we have another random variable $Z$ on $\mathcal{Z}$ and RKHS $(\mathcal{H}_\mathcal{Z}, k_\mathcal{Z})$, which satisfy the analog to (A-1). We then define the *normalized conditional cross-covariance operator*,

$$V_{YX|Z} = V_{YX} - V_{YZ}V_{ZX}, \tag{3}$$

for measuring the conditional dependence of $X$ and $Y$ given $Z$, where $V_{YZ}$ and $V_{ZX}$ are defined similarly to Eq. (2). The operator $V_{YX|Z}$ may be better understood by expressing it as

$$V_{YX|Z} = \Sigma_{YY}^{-1/2} \big( \Sigma_{YX} - \Sigma_{YZ}\Sigma_{ZZ}^{-1}\Sigma_{ZX} \big) \Sigma_{XX}^{-1/2},$$

where $\Sigma_{YX|Z} = \Sigma_{YX} - \Sigma_{YZ}\Sigma_{ZZ}^{-1}\Sigma_{ZX}$ can be interpreted as a nonlinear extension of the conditional covariance matrix $C_{YX} - C_{YZ}C_{ZZ}^{-1}C_{ZX}$ of Gaussian random variables.

The operator $\Sigma_{YX}$ can be used to determine the independence of $X$ and $Y$: roughly speaking, $\Sigma_{YX} = O$ if and only if $X \perp\!\!\!\perp Y$. Similarly, a relation between $\Sigma_{YX|Z}$ and conditional independence, $X \perp\!\!\!\perp Y \mid Z$, has been established in [5]: if the extended variables $\ddot{X} = (X, Z)$ and $\ddot{Y} = (Y, Z)$ are used, $X \perp\!\!\!\perp Y \mid Z$ is equivalent to $\Sigma_{\ddot{X}\ddot{Y}|Z} = O$. We will give a rigorous treatment in Section 2.2

Noting that the conditions $\Sigma_{YX} = O$ and $\Sigma_{YX|Z} = O$ are equivalent to $V_{YX} = O$ and $V_{YX|Z} = O$, respectively, we propose to use the Hilbert-Schmidt norms of the latter operators as dependence

measures. Recall that an operator $A : \mathcal{H}_1 \to \mathcal{H}_2$ is called Hilbert-Schmidt if for complete orthonormal systems (CONSs) $\{\phi_i\}$ of $\mathcal{H}_1$ and $\{\psi_j\}$ of $\mathcal{H}_2$, the sum $\sum_{i,j}\langle\psi_j, A\phi_i\rangle^2_{\mathcal{H}_2}$ is finite (see [13]). For a Hilbert-Schmidt operator $A$, the Hilbert-Schmidt (HS) norm $\|A\|_{HS}$ is defined by $\|A\|^2_{HS} = \sum_{i,j}\langle\psi_j, A\phi_i\rangle^2_{\mathcal{H}_2}$. It is easy to see that this sum is independent of the choice of CONSs. Provided that $V_{YX}$ and $V_{YX|Z}$ are Hilbert-Schmidt, we propose the following measures:

$$I^{COND}(X,Y|Z) = \|V_{\ddot{Y}\ddot{X}|Z}\|^2_{HS}, \tag{4}$$

$$I^{NOCCO}(X,Y) = \|V_{YX}\|^2_{HS}. \tag{5}$$

A sufficient condition that these operators are Hilbert-Schmidt will be discussed in Section 2.3.

It is easy to provide empirical estimates of the measures. Let $(X_1,Y_1,Z_1),\ldots,(X_n,Y_n,Z_n)$ be an i.i.d. sample from the joint distribution. Using the empirical mean elements $\widehat{m}_X^{(n)} = \frac{1}{n}\sum_{i=1}^n k_{\mathcal{X}}(\,\cdot\,,X_i)$ and $\widehat{m}_Y^{(n)} = \frac{1}{n}\sum_{i=1}^n k_{\mathcal{Y}}(\,\cdot\,,Y_i)$, an estimator of $\Sigma_{YX}$ is

$$\widehat{\Sigma}_{YX}^{(n)} = \tfrac{1}{n}\sum_{i=1}^n (k_{\mathcal{Y}}(\,\cdot\,,Y_i) - \widehat{m}_Y^{(n)})\langle k_{\mathcal{X}}(\,\cdot\,,X_i) - \widehat{m}_X^{(n)}, \,\cdot\,\rangle_{\mathcal{H}_{\mathcal{X}}}.$$

$\widehat{\Sigma}_{XX}^{(n)}$ and $\widehat{\Sigma}_{YY}^{(n)}$ are defined similarly. The estimators of $V_{YX}$ and $V_{YX|Z}$ are respectively

$$\widehat{V}_{YX}^{(n)} = \big(\widehat{\Sigma}_{YY}^{(n)} + \varepsilon_n I\big)^{-1/2}\widehat{\Sigma}_{YX}^{(n)}\big(\widehat{\Sigma}_{XX}^{(n)} + \varepsilon_n I\big)^{-1/2},$$

where $\varepsilon_n > 0$ is a regularization constant used in the same way as [1, 5], and

$$\widehat{V}_{YX|Z}^{(n)} = \widehat{V}_{YX}^{(n)} - \widehat{V}_{YZ}^{(n)}\widehat{V}_{ZX}^{(n)}, \tag{6}$$

from Eq. (3). The HS norm of the finite rank operator $\widehat{V}_{YX|Z}^{(n)}$ is easy to calculate. Let $G_X$, $G_Y$, and $G_Z$ be the centered Gram matrices, such that $G_{X,ij} = \langle k_{\mathcal{X}}(\,\cdot\,,X_i) - \widehat{m}_X^{(n)}, k_{\mathcal{X}}(\,\cdot\,,X_j) - \widehat{m}_X^{(n)}\rangle_{\mathcal{H}_{\mathcal{X}}}$ and so on, and define $R_X$, $R_Y$, and $R_Z$ as $R_X = G_X(G_X + n\varepsilon_n I_n)^{-1}$, $R_Y = G_Y(G_Y + n\varepsilon_n I_n)^{-1}$, and $R_Z = G_Z(G_Z + n\varepsilon_n I_n)^{-1}$. The empirical dependence measures are then

$$\hat{I}_n^{COND} \equiv \big\|\widehat{V}_{\ddot{Y}\ddot{X}|Z}^{(n)}\big\|^2_{HS} = \mathrm{Tr}\big[R_{\ddot{Y}}R_{\ddot{X}} - 2R_{\ddot{Y}}R_{\ddot{X}}R_Z + R_{\ddot{Y}}R_Z R_{\ddot{X}}R_Z\big], \tag{7}$$

$$\hat{I}_n^{NOCCO}(X,Y) \equiv \big\|\widehat{V}_{YX}^{(n)}\big\|^2_{HS} = \mathrm{Tr}\big[R_Y R_X\big], \tag{8}$$

where the extended variables are used for $\hat{I}_n^{COND}$. These empirical estimators, and use of $\varepsilon_n$, will be justified in Section 2.4 by showing the convergence to $I^{NOCCO}$ and $I^{COND}$. With the incomplete Cholesky decomposition [17] of rank $r$, the complexity to compute $\hat{I}_n^{COND}$ is $O(r^2 n)$.

## 2.2 Inference on probabilities by characteristic kernels

To relate $I^{NOCCO}$ and $I^{COND}$ with independence and conditional independence, respectively, the RKHS should contain a sufficiently rich class of functions to represent all higher order moments. Similar notions have already appeared in the literature: universal kernel on compact domains [15] and Gaussian kernels on the entire $\mathbb{R}^m$ characterize independence via the cross-covariance operator [8, 1]. We now discuss a unified class of kernels for inference on probabilities.

Let $(\mathcal{X},\mathcal{B})$ be a measurable space, $X$ a random variable on $\mathcal{X}$, and $(\mathcal{H},k)$ an RKHS on $\mathcal{X}$ satisfying assumption (A-1). The *mean element* of $X$ on $\mathcal{H}$ is defined by the unique element $m_X \in \mathcal{H}$ such that $\langle m_X, f\rangle_{\mathcal{H}} = E[f(X)]$ for all $f \in \mathcal{H}$ (see [7]). If the distribution of $X$ is $P$, we also use $m_P$ to denote $m_X$. Letting $\mathcal{P}$ be the family of all probabilities on $(\mathcal{X},\mathcal{B})$, we define the map $\mathcal{M}_k$ by

$$\mathcal{M}_k : \mathcal{P} \to \mathcal{H}, \qquad P \mapsto m_P.$$

The kernel $k$ is said to be *characteristic*[1] if the map $\mathcal{M}_k$ is injective, or equivalently, if the condition $E_{X\sim P}[f(X)] = E_{X\sim Q}[f(X)]$ $(\forall f \in \mathcal{H})$ implies $P = Q$.

The notion of a characteristic kernel is an analogy to the characteristic function $E_P[e^{\sqrt{-1}u^T X}]$, which is the expectation of the Fourier kernel $k_F(x,u) = e^{\sqrt{-1}u^T x}$. Noting that $m_P = m_Q$ iff $E_P[k(u,X)] = E_Q[k(u,X)]$ for all $u \in \mathcal{X}$, the definition of a characteristic kernel generalizes the well-known property of the characteristic function that $E_P[k_F(u,X)]$ uniquely determines a Borel probability $P$ on $\mathbb{R}^m$. The next lemma is useful to show that a kernel is characteristic.

**Lemma 1.** *Let $q \geq 1$. Suppose that $(\mathcal{H}, k)$ is an RKHS on a measurable space $(\mathcal{X}, \mathcal{B})$ with $k$ measurable and bounded. If $\mathcal{H} + \mathbb{R}$ (the direct sum of the two RKHSs) is dense in $L^q(\mathcal{X}, P)$ for any probability $P$ on $(\mathcal{X}, \mathcal{B})$, the kernel $k$ is characteristic.*

*Proof.* Assume $m_P = m_Q$. By the assumption, for any $\varepsilon > 0$ and a measurable set $A$, there is a function $f \in \mathcal{H}$ and $c \in \mathbb{R}$ such that $|E_P[f(X)] + c - P(A)| < \varepsilon$ and $|E_Q[f(Y)] + c - Q(A)| < \varepsilon$, from which we have $|P(A) - Q(A)| < 2\varepsilon$. Since $\varepsilon > 0$ is arbitrary, this means $P(A) = Q(A)$. $\square$

Many popular kernels are characteristic. For a compact metric space, it is easy to see that the RKHS given by a universal kernel [15] is dense in $L^2(P)$ for any $P$, and thus characteristic (see also [7] Theorem 3). It is also important to consider kernels on non-compact spaces, since many standard random variables, such as Gaussian variables, are defined on non-compact spaces. The next theorem implies that many kernels on the entire $\mathbb{R}^m$, including Gaussian and Laplacian, are characteristic. The proof is an extension of Theorem 2 in [1], and is given in the supplementary material.

**Theorem 2.** *Let $\phi(z)$ be a continuous positive function on $\mathbb{R}^m$ with the Fourier transform $\tilde{\phi}(u)$, and $k$ be a kernel of the form $k(x, y) = \phi(x - y)$. If for any $\xi \in \mathbb{R}^m$ there exists $\tau_0$ such that $\int \frac{\tilde{\phi}(\tau(u+\xi))^2}{\tilde{\phi}(u)} du < \infty$ for all $\tau > \tau_0$, then the RKHS associated with $k$ is dense in $L^2(P)$ for any Borel probability $P$ on $\mathbb{R}^m$. Hence $k$ is characteristic with respect to the Borel $\sigma$-field.*

The assumptions to relate the operators with independence are well described by using characteristic kernels and denseness. The next result generalizes Corollary 9 in [5] (we omit the proof: see [5, 6]).

**Theorem 3.** *(i) Assume (A-1) for the kernels. If the product $k_\mathcal{X} k_\mathcal{Y}$ is characteristic, then we have*

$$V_{YX} = O \qquad \Longleftrightarrow \qquad X \perp\!\!\!\perp Y.$$

*(ii) Denote $\ddot{X} = (X, Z)$ and $k_{\ddot{X}} = k_\mathcal{X} k_\mathcal{Z}$. In addition to (A-1), assume that the product $k_{\ddot{X}} k_\mathcal{Y}$ is a characteristic kernel on $(\mathcal{X} \times \mathcal{Z}) \times \mathcal{Y}$, and $\mathcal{H}_\mathcal{Z} + \mathbb{R}$ is dense in $L^2(P_\mathcal{Z})$. Then,*

$$V_{Y\ddot{X}|Z} = O \qquad \Longleftrightarrow \qquad X \perp\!\!\!\perp Y \mid Z.$$

From the above results, we can guarantee that $V_{YX}$ and $V_{Y\ddot{X}|Z}$ will detect independence and conditional independence, if we use a Gaussian or Laplacian kernel either on a compact set or the whole of $\mathbb{R}^m$. Note also that we can substitute $V_{\ddot{Y}\ddot{X}|Z}$ for $V_{Y\ddot{X}|Z}$ in Theorem 3 *(ii)*.

### 2.3 Kernel-free integral expression of the measures

A remarkable property of $I^{NOCCO}$ and $I^{COND}$ is that they do not depend on the kernels under some assumptions, having integral expressions containing only the probability density functions. The probability $E_Z[P_{X|Z} \otimes P_{Y|Z}]$ on $\mathcal{X} \times \mathcal{Y}$ is defined by $E_Z[P_{Y|Z} \otimes P_{X|Z}](B \times A) \int E[\chi_B(Y)|Z = z] E[\chi_A(X)|Z = z] dP_Z(z)$ for $A \in \mathcal{B}_\mathcal{X}$ and $B \in \mathcal{B}_\mathcal{Y}$.

**Theorem 4.** *Let $\mu_\mathcal{X}$ and $\mu_\mathcal{Y}$ be measures on $\mathcal{X}$ and $\mathcal{Y}$, respectively, and assume that the probabilities $P_{XY}$ and $E_Z[P_{X|Z} \otimes P_{Y|Z}]$ are absolutely continuous with respect to $\mu_\mathcal{X} \times \mu_\mathcal{Y}$ with probability density functions $p_{XY}$ and $p_{X \perp\!\!\!\perp Y|Z}$, respectively. If $\mathcal{H}_\mathcal{Z} + \mathbb{R}$ and $(\mathcal{H}_\mathcal{X} \otimes \mathcal{H}_\mathcal{Y}) + \mathbb{R}$ are dense in $L^2(P_Z)$ and $L^2(P_X \otimes P_Y)$, respectively, and $V_{YX}$ and $V_{YZ} V_{ZX}$ are Hilbert-Schmidt, then we have*

$$I^{COND} = \|V_{YX|Z}\|_{HS}^2 = \int\!\!\!\int_{\mathcal{X} \times \mathcal{Y}} \left( \frac{p_{XY}(x, y)}{p_X(x) p_Y(y)} - \frac{p_{X \perp\!\!\!\perp Y|Z}(x, y)}{p_X(x) p_Y(y)} \right)^2 p_X(x) p_Y(y) d\mu_\mathcal{X} d\mu_\mathcal{Y},$$

*where $p_X$ and $p_Y$ are the density functions of the marginal distributions $P_X$ and $P_Y$, respectively. As a special case of $\mathcal{Z} = \emptyset$, we have*

$$I^{NOCCO} = \|V_{YX}\|_{HS}^2 = \int\!\!\!\int_{\mathcal{X} \times \mathcal{Y}} \left( \frac{p_{XY}(x, y)}{p_X(x) p_Y(y)} - 1 \right)^2 p_X(x) p_Y(y) d\mu_\mathcal{X} d\mu_\mathcal{Y}. \qquad (9)$$

*Sketch of the proof (see the supplement for the complete proof).* Since it is known [8] that $\Sigma_{ZZ}$ is Hilbert-Schmidt under (A-1), there exist CONSs $\{\phi_i\}_{i=1}^\infty \subset \mathcal{H}_\mathcal{X}$ and $\{\psi_j\}_{j=1}^\infty \subset \mathcal{H}_\mathcal{Y}$ consisting of the eigenfunctions of $\Sigma_{XX}$ and $\Sigma_{YY}$, respectively, with $\Sigma_{XX} \phi_i = \lambda_i \phi_i$ ($\lambda_i \geq 0$) and $\Sigma_{YY} \psi_j =$

$\nu_j \psi_j$ $(\nu_j \geq 0)$. Then, $\|V_{YX|Z}\|_{HS}^2$ admits the expansion

$$\sum_{i,j=1}^{\infty} \left\{ \langle \psi_j, V_{YX}\phi_i \rangle_{\mathcal{H}_{\mathcal{Y}}}^2 - 2\langle \psi_j, V_{YX}\phi_i \rangle_{\mathcal{H}_{\mathcal{Y}}} \langle \psi_j, V_{YZ}V_{ZX}\phi_i \rangle_{\mathcal{H}_{\mathcal{Y}}} + \langle \psi_j, V_{YZ}V_{ZX}\phi_i \rangle_{\mathcal{H}_{\mathcal{Y}}}^2 \right\}.$$

Let $I_+^X = \{i \in \mathbb{N} \mid \lambda_i > 0\}$ and $I_+^Y = \{i \in \mathbb{N} \mid \nu_i > 0\}$, and define $\tilde{\phi}_i = (\phi_i - E[\phi_i(X)])/\sqrt{\lambda_i}$ and $\tilde{\psi}_j = (\psi_j - E[\psi_j(Y)])/\sqrt{\nu_j}$ for $i \in I_+^X$ and $j \in I_+^Y$. For simplicity, $L^2$ denotes $L_2(P_X \otimes P_Y)$. With the notations $\tilde{\phi}_0 = 1$ and $\tilde{\psi}_0 = 1$, it is easy to see that the class $\{\tilde{\phi}_i\tilde{\psi}_j\}_{i \in I_+^X \cup \{0\}, j \in I_+^Y \cup \{0\}}$ is a CONS of $L^2$. From Parseval's equality, the first term of the above expansion is rewritten as

$$\sum_{i \in I_+^X, j \in I_+^Y} \langle \tilde{\psi}_j, \Sigma_{YX}\tilde{\phi}_i \rangle_{\mathcal{H}_{\mathcal{Y}}}^2 = \sum_{i \in I_+^X, j \in I_+^Y} E_{YX}\left[\tilde{\psi}_j(Y)\tilde{\phi}_i(X)\right]^2 = \sum_{i \in I_+^X, j \in I_+^Y} \left(\tilde{\phi}_i\tilde{\psi}_j, \frac{p_{XY}}{p_X p_Y}\right)_{L^2}$$

$$= \left\|\frac{p_{XY}(x,y)}{p_X(x)p_Y(y)}\right\|_{L_2}^2 - \sum_{i \in I_+^X} E\left[\tilde{\phi}_i(X)\right] - \sum_{j \in I_+^Y} E\left[\tilde{\psi}_j(Y)\right] - 1 = \left\|\frac{p_{XY}(x,y)}{p_X(x)p_Y(y)}\right\|_{L_2}^2 - 1.$$

By a similar argument, the second and third term of the expansion are rewritten as $-2\left(\frac{p_{XY}}{p_X p_Y}, \frac{p_{X \perp\!\!\!\perp Y|Z}}{p_X p_Y}\right)_{L_2} + 2$ and $\left\|\frac{p_{X \perp\!\!\!\perp Y|Z}}{p_X p_Y}\right\|_{L_2}^2 - 1$, respectively. This completes the proof. $\qquad\square$

Many practical kernels, such as the Gaussian and Laplacian, satisfy the assumptions in the above theorem, as we saw in Theorems 2 and the remark after Lemma 1. While the empirical estimate from finite samples depends on the choice of kernels, it is a desirable property for the empirical dependence measure to converge to a value that depends only on the distributions of the variables.

Eq. (9) shows that, under the assumptions, $I^{NOCCO}$ is equal to the *mean square contingency*, a well-known dependence measure[14] commonly used for discrete variables. As we show in Section 2.4, $\hat{I}_n^{NOCCO}$ works as a consistent kernel estimator of the mean square contingency.

The expression of Eq. (9) can be compared with the mutual information,

$$MI(X,Y) = \int\!\!\int_{\mathcal{X}\times\mathcal{Y}} p_{XY}(x,y) \log \frac{p_{XY}(x,y)}{p_X(x)p_Y(y)} d\mu_{\mathcal{X}} d\mu_{\mathcal{Y}}.$$

Both the mutual information and the mean square contingency are nonnegative, and equal to zero if and only if $X$ and $Y$ are independent. Note also that from $\log z \leq z - 1$, the inequality $MI(X,Y) \leq I^{NOCCO}(X,Y)$ holds under the assumptions of Theorem 4. While the mutual information is the best known dependence measure, its finite sample empirical estimate is not straightforward, especially for continuous variables. The direct estimation of a probability density function is infeasible if the joint space has even a moderate number of dimensions.

## 2.4 Consistency of the measures

It is important to ask whether the empirical measures converge to the population value $I^{COND}$ and $I^{NOCCO}$, since this provides a theoretical justification for the empirical measures. It is known [4] that $\widehat{V}_{YX}^{(n)}$ converges in probability to $V_{YX}$ in operator norm. The next theorem asserts convergence in HS norm, provided that $V_{YX}$ is Hilbert-Schmidt. Although the proof is analogous to the case of operator norm, it is more involved to discuss the HS norm. We give it in the supplementary material.

**Theorem 5.** *Assume that $V_{YX}$, $V_{YZ}$, and $V_{ZX}$ are Hilbert-Schmidt, and that the regularization constant $\varepsilon_n$ satisfies $\varepsilon_n \to 0$ and $\varepsilon_n^3 n \to \infty$. Then, we have the convergence in probability*

$$\|\widehat{V}_{YX}^{(n)} - V_{YX}\|_{HS} \to 0 \qquad and \qquad \|\widehat{V}_{YX|Z}^{(n)} - V_{YX|Z}\|_{HS} \to 0 \qquad (n \to \infty). \qquad (10)$$

*In particular, $\hat{I}_n^{NOCCO} \to I^{NOCCO}$ and $\hat{I}_n^{COND} \to I^{COND}$ $(n \to \infty)$ in probability.*

## 2.5 Choice of kernels

As with all empirical measures, the sample estimates $\hat{I}_n^{NOCCO}$ and $\hat{I}_n^{COND}$ are dependent on the kernel, and the problem of choosing a kernel has yet to be solved. Unlike supervised learning, there are no easy criteria to choose a kernel for dependence measures. We propose a method of choosing a kernel by considering the large sample behavior. We explain the method only briefly in this paper.

The basic idea is that a kernel should be chosen so that the covariance operator detects independence of variables as effectively as possible. It has been recently shown [10], under the independence of

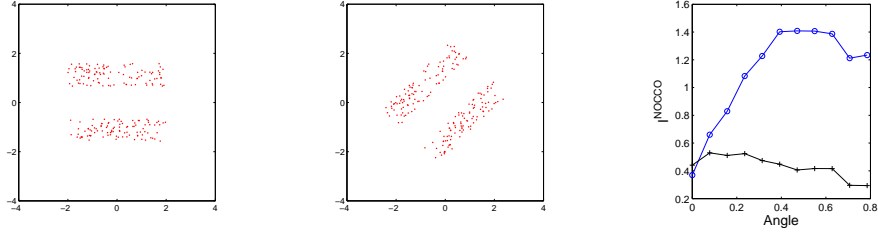

Figure 1: Left and Middle: Examples of data ($\theta = 0$ and $\theta = \pi/4$). Right: The marks "o" and "+" show $\hat{I}_n^{NOCCO}$ for each angle and the 95th percentile of the permutation test, respectively.

$X$ and $Y$, that the measure $HSIC = \|\widehat{\Sigma}_{YX}^{(n)}\|_{HS}^2$ ([8]) multiplied by $n$ converges to an infinite mixture of $\chi^2$ distributions with variance $\mathrm{Var}^{lim}[nHSIC] = 2\|\Sigma_{XX}\|_{HS}^2\|\Sigma_{YY}\|_{HS}^2$. We choose a kernel so that the bootstrapped variance $\mathrm{Var}^B[nHSIC]$ of $nHSIC$ is close to this theoretical limit variance. More precisely, we compare the ratio $T = \mathrm{Var}^B[nHSIC]/\mathrm{Var}^{lim}[nHSIC]$ for various candidate kernels. In preliminary experiments for choosing the variance parameter $\sigma$ of Gaussian kernels, we often observed the ratio decays and saturates below 1, as $\sigma$ increases. Therefore, we use $\sigma$ starting the saturation by choosing the minimum of $\sigma$ among all candidates that satisfy $|T_\sigma - \alpha| \leq (1 + \delta)\min_\sigma |T_\sigma - \alpha|$ for $\delta > 0, \alpha \in (0, 1]$. We always use $\delta = 0.1$ and $\alpha = 0.5$. We can expect that the chosen kernel uses the data effectively. While there is no rigorous theoretical guarantee, in the next section we see that the method gives a reasonable result for $\hat{I}_n^{NOCCO}$ and $\hat{I}_n^{COND}$.

## 3  Experiments

To evaluate the dependence measures, we use a permutation test of independence for data sets with various degrees of dependence. The test randomly permutes the order of $Y_1, \ldots, Y_n$ to make many samples independent of $(X_1, \ldots, X_n)$, thus simulating the null distribution under independence. For the evaluation of $\hat{I}_n^{COND}$, the range of $Z$ is partitioned into $\mathcal{Z}_1, \ldots, \mathcal{Z}_L$ with the same number of data, and the sample $\{(X_i, Y_i) \mid Z_i \in \mathcal{Z}_\ell\}$ within the $\ell$-th bin is randomly permuted. The significance level is always set to $5\%$. In the following experiments, we always use Gaussian kernels $e^{-\frac{1}{2\sigma^2}\|x_1 - x_2\|^2}$ and choose $\sigma$ by the method proposed in Section 2.5.

**Synthetic data for dependence.** The random variables $X^{(0)}$ and $Y^{(0)}$ are independent and uniformly distributed on $[-2, 2]$ and $[a, b] \cup [-b, -a]$, respectively, so that $(X^{(0)}, Y^{(0)})$ has a scalar covariance matrix. $(X^{(\theta)}, Y^{(\theta)})$ is the rotation of $(X^{(0)}, Y^{(0)})$ by $\theta \in [0, \pi/4]$ (see Figure 1). $X^{(\theta)}$ and $Y^{(\theta)}$ are always uncorrelated, but dependent for $\theta \neq 0$. We generate 100 sets of 200 data. We perform permutation tests with $\hat{I}_n^{NOCCO}$, $HSIC = \|\widehat{\Sigma}_{YX}^{(n)}\|_{HS}^2$, and the mutual information (MI). For the empirical estimates of MI, we use the advanced method from [11], with no need for explicit estimation of the densities. Since $\hat{I}_n^{NOCCO}$ is an estimate of the mean square contingency, we also apply a relevant contingency-table-based independence test ([12]), partitioning the variables into bins. Figure 1 shows the values of $\hat{I}_n^{NOCCO}$ for a sample. In Table 1, we see that the results of $\hat{I}_n^{NOCCO}$ are stable w.r.t. the choice of $\varepsilon_n$, provided it is sufficiently small. We fix $\varepsilon_n = 10^{-6}$ for all remaining experiments. While all the methods are able to detect the dependence, $\hat{I}_n^{NOCCO}$ with the asymptotic choice of $\sigma$ is the most sensitive to very small dependence. We also observe the chosen parameters $\sigma_Y$ for $Y$ increase from 0.58 to 2.0 as $\theta$ increases. The small $\sigma_Y$ for small $\theta$ seems reasonable, because the range of $Y$ is split into two small regions.

**Chaotic time series.** We evaluate a chaotic time series derived from the coupled Hénon map. The variables $X$ and $Y$ are four dimensional: the components $X_1, X_2, Y_1$, and $Y_2$ follow the dynamics $(X_1(t + 1), X_2(t + 1)) = (1.4 - X_1(t)^2 + 0.3X_2(t), X_1(t))$, $(Y_1(t + 1), Y_2(t + 1)) = (1.4 - \{\gamma X_1(t)Y_1(t) + (1 - \gamma)Y_2(t)^2\} + 0.1Y_2(t), Y_1(t))$, and $X_3, X_4, Y_3, Y_4$ are independent noise with $N(0, (0.5)^2)$. $X$ and $Y$ are independent for $\gamma = 0$, while they are synchronized chaos for $\gamma > 0$ (see Figure 2 for examples). A sample consists of 100 data generated from this system. Table 2

| Angle (degree) | 0 | 4.5 | 9 | 13.5 | 18 | 22.5 | 27 | 31.5 | 36 | 40.5 | 45 |
|---|---|---|---|---|---|---|---|---|---|---|---|
| $\hat{I}_n^{NOCCO}$ ($\varepsilon = 10^{-4}$, Median) | 94 | 23 | 0 | 0 | 0 | 0 | 0 | 0 | 0 | 0 | 0 |
| $\hat{I}_n^{NOCCO}$ ($\varepsilon = 10^{-6}$, Median) | 92 | 20 | 1 | 0 | 0 | 0 | 0 | 0 | 0 | 0 | 0 |
| $\hat{I}_n^{NOCCO}$ ($\varepsilon = 10^{-8}$, Median) | 93 | 15 | 0 | 0 | 0 | 0 | 0 | 0 | 0 | 0 | 0 |
| $\hat{I}_n^{NOCCO}$ (Asymp. Var.) | 94 | 11 | 0 | 0 | 0 | 0 | 0 | 0 | 0 | 0 | 0 |
| HSIC (Median) | 93 | 92 | 63 | 5 | 0 | 0 | 0 | 0 | 0 | 0 | 0 |
| HSIC (Asymp. Var.) | 93 | 44 | 1 | 0 | 0 | 0 | 0 | 0 | 0 | 0 | 0 |
| MI (#Nearest Neighbors = 1) | 93 | 62 | 11 | 0 | 0 | 0 | 0 | 0 | 0 | 0 | 0 |
| MI (#Nearest Neighbors = 3) | 96 | 43 | 0 | 0 | 0 | 0 | 0 | 0 | 0 | 0 | 0 |
| MI (#Nearest Neighbors = 5) | 97 | 49 | 0 | 0 | 0 | 0 | 0 | 0 | 0 | 0 | 0 |
| Conting. Table (#Bins= 3) | 100 | 96 | 46 | 9 | 1 | 0 | 0 | 0 | 0 | 0 | 0 |
| Conting. Table (#Bins= 4) | 98 | 29 | 0 | 0 | 0 | 0 | 0 | 0 | 0 | 0 | 0 |
| Conting. Table (#Bins= 5) | 98 | 82 | 5 | 0 | 0 | 0 | 0 | 0 | 0 | 0 | 0 |

Table 1: Comparison of dependence measures. The number of times independence is accepted out of 100 permutation tests is shown. "Asymp. Var." is the method in Section 2.5. "Median" is a heuristic method [8] which chooses $\sigma$ as the median of pairwise distances of the data.

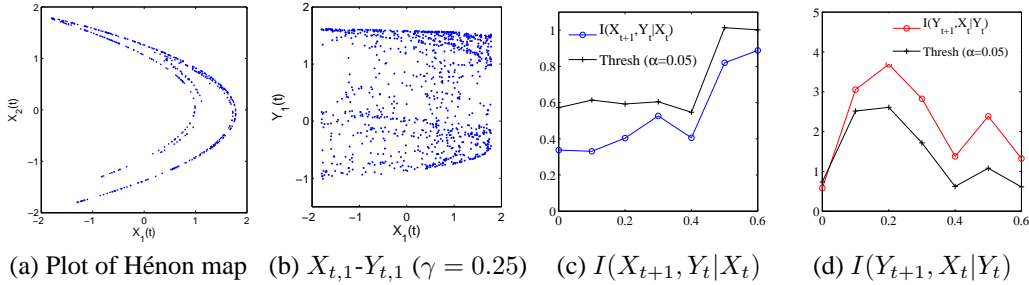

(a) Plot of Hénon map   (b) $X_{t,1}$-$Y_{t,1}$ ($\gamma = 0.25$)   (c) $I(X_{t+1}, Y_t | X_t)$   (d) $I(Y_{t+1}, X_t | Y_t)$

Figure 2: Chaotic time series. (a,b): examples of data. (c,d) examples of $\hat{I}_n^{COND}$ (colored "o") and the threshholds of the permutation test with significance level $5\%$ (black "+").

shows the results of permutation tests of independence for the instantaneous pairs $(X(t), Y(t))_{t=1}^{100}$. The proposed $\hat{I}_n^{NOCCO}$ outperforms the other methods in capturing small dependence.

Next, we apply $\hat{I}_n^{COND}$ to detect the causal structure of the same time series. Note that the series $X$ is a cause of $Y$ for $\gamma > 0$, but there is no opposite causality, i.e., $X_{t+1} \perp\!\!\!\perp Y_t \,|\, X_t$ and $Y_{t+1} \not\!\perp\!\!\!\perp X_t \,|\, Y_t$. In Table 3, it is remarkable that $\hat{I}_n^{COND}$ detects the small causal influence from $X_t$ to $Y_{t+1}$ for $\gamma \geq 0.1$, while for $\gamma = 0$ the result is close to the theoretical value of 95%.

**Graphical modeling from medical data.** This is the inference of a graphical model from data with no time structure. The data consist of three variables, creatinine clearance (C), digoxin clearance (D), urine flow (U). These were taken from 35 patients, and analyzed with graphical models in [3, Section 3.1.4.]. From medical knowledge, D should be independent of U when controlling C. Table 4 shows the results of the permutation tests and a comparison with the linear method. The relation $D \perp\!\!\!\perp U \,|\, C$ is strongly affirmed by $\hat{I}_n^{COND}$, while the partial correlation does not find it.

| $\gamma$ (strength of coupling) | 0.0 | 0.1 | 0.2 | 0.3 | 0.4 | 0.5 | 0.6 |
|---|---|---|---|---|---|---|---|
| $\hat{I}_n^{NOCCO}$ | 97 | 66 | 21 | 1 | 0 | 1 | 0 |
| HSIC | 75 | 70 | 58 | 52 | 13 | 1 | 0 |
| MI ($k = 3$) | 87 | 91 | 83 | 73 | 23 | 6 | 0 |
| MI ($k = 5$) | 87 | 88 | 75 | 67 | 23 | 5 | 0 |
| MI ($k = 7$) | 87 | 86 | 75 | 64 | 21 | 5 | 0 |

Table 2: Results for the independence tests for the chaotic time series. The number of times independence was accepted out of 100 permutation tests is shown. $\gamma = 0$ implies independence.

| $\gamma$ (coupling) | $H_0$: $Y_t$ is *not* a cause of $X_{t+1}$ | | | | | | | $H_0$: $X_t$ is *not* a cause of $Y_{t+1}$ | | | | | | |
|---|---|---|---|---|---|---|---|---|---|---|---|---|---|---|
| | 0.0 | 0.1 | 0.2 | 0.3 | 0.4 | 0.5 | 0.6 | 0.0 | 0.1 | 0.2 | 0.3 | 0.4 | 0.5 | 0.6 |
| $\hat{I}_n^{NOCCO}$ | 97 | 96 | 93 | 85 | 81 | 68 | 75 | 96 | 0 | 0 | 0 | 0 | 0 | 0 |
| HSIC | 94 | 94 | 92 | 81 | 60 | 73 | 66 | 93 | 95 | 85 | 56 | 1 | 1 | 1 |

Table 3: Results of the permutation test of *non-causality* for the chaotic time series. The number of times non-causality was accepted out of 100 tests is shown.

| | Kernel measure | | | Linear method | | |
|---|---|---|---|---|---|---|
| | $\hat{I}_n^{COND}$ | $P$-value | | | (partial) correl. | $P$-value |
| $D \perp\!\!\!\perp U \mid C$ | 1.458 | 0.924 | | $\mathrm{Parcorr}(D,U|C)$ | 0.4847 | 0.0037 |
| $C \perp\!\!\!\perp D$ | 0.776 | <0.001 | | $\mathrm{Corr}(C,D)$ | 0.7754 | 0.0000 |
| $C \perp\!\!\!\perp U$ | 0.194 | 0.117 | | $\mathrm{Corr}(C,U)$ | 0.3092 | 0.0707 |
| $D \perp\!\!\!\perp U$ | 0.343 | 0.023 | | $\mathrm{Corr}(D,U)$ | 0.5309 | 0.0010 |

Table 4: Graphical modeling from the medical data. Higher $P$-values indicate (conditional) independence more strongly.

# 4   Concluding remarks

There are many dependence measures, and further theoretical and experimental comparison is important. That said, one unambiguous strength of the kernel measure we propose is its kernel-free population expression. It is interesting to ask if other classical dependence measures, such as the mutual information, can be estimated by kernels (in a broader sense than the expansion about independence of [9]). A relevant measure is the kernel generalized variance (KGV [1]), which is based on a sum of the logarithm of the eigenvalues of $V_{YX}$, while $I^{NOCCO}$ is their squared sum. It is also interesting to investigate whether the KGV has a kernel-free expression. Another topic for further study is causal inference with the proposed measure, both with and without time information ([16]).

## Footnotes

[1] Although the same notion was called *probability-determining* in [5], we call it "characteristic" by analogy with the characteristic function.

# References

[1] F. Bach and M. Jordan. Kernel independent component analysis. *J. Machine Learning Res.*, 3:1–48, 2002.

[2] C. Baker. Joint measures and cross-covariance operators. *Trans. Amer. Math. Soc.*, 186:273–289, 1973.

[3] D. Edwards. *Introduction to graphical modelling*. Springer verlag, New York, 2000.

[4] K. Fukumizu, F. Bach, and A. Gretton. Statistical consistency of kernel canonical correlation analysis. *J. Machine Learning Res.*, 8:361–383, 2007.

[5] K. Fukumizu, F. Bach, and M. Jordan. Dimensionality reduction for supervised learning with reproducing kernel Hilbert spaces. *J. Machine Learning Res.*, 5:73–99, 2004.

[6] K. Fukumizu, F. Bach, and M. Jordan. Kernel dimension reduction in regression. Tech Report 715, Dept. Statistics, University of California, Berkeley, 2006.

[7] A. Gretton, K. Borgwardt, M. Rasch, B. Schölkopf, and A. Smola. A kernel method for the two-sample-problem. *Advances in NIPS 19*. MIT Press, 2007.

[8] A. Gretton, O. Bousquet, A. Smola, and B. Schölkopf. Measuring statistical dependence with Hilbert-Schmidt norms. *16th Intern. Conf. Algorithmic Learning Theory*, pp.63–77. Springer, 2005.

[9] A. Gretton, R. Herbrich, A. Smola, O. Bousquet and B. Schölkopf. Kernel Methods for Measuring Independence. *J. Machine Learning Res.*, 6:2075–2129, 2005.

[10] A. Gretton, K. Fukumizu, C. Teo, L. Song, B. Schölkopf, A. Smola. A Kernel Statistical Test of Independence. Advances in NIPS 21. 2008, to appear.

[11] A. Kraskov, H. Stögbauer, and P. Grassberger. Estimating mutual information. *Physical Review E*, 69, 066138-1–16, 2004.

[12] T. Read and N. Cressie. *Goodness-of-Fit Statistics for Discrete Multivariate Data*. Springer-Verlag, 1988.

[13] M. Reed and B. Simon. *Functional Analysis*. Academic Press, 1980.

[14] A. Rényi. *Probability Theory*. Horth-Holland, 1970.

[15] I. Steinwart. On the influence of the kernel on the consistency of support vector machines. *J. Machine Learning Res.*, 2:67–93, 2001.

[16] X. Sun, D. Janzing, B. Schölkopf, and K. Fukumizu. A kernel-based causal learning algorithm. *Proc. 24th Intern. Conf. Machine Learning*, 2007 to appear.

[17] S. Fine and K. Scheinberg Efficient SVM Training using Low-Rank Kernel Representations *J. Machine Learning Res.*, 2:243–264, 2001.

